# Unsupervised Color Decomposition of Histologically Stained Tissue Samples

**A. Rabinovich**
Department of Computer Science
University of California, San Diego
amrabino@ucsd.edu

**S. Agarwal**
Department of Computer Science
University of California, San Diego
sagarwal@cs.ucsd.edu

**C. A. Laris**
Q3DM, Inc.
claris@q3dm.com

**J.H. Price**
Department of Bioengineering
University of California, San Diego
jhprice@ucsd.edu

**S. Belongie**
Department of Computer Science
University of California, San Diego
sjb@cs.ucsd.edu

## Abstract

Accurate spectral decomposition is essential for the analysis and diagnosis of histologically stained tissue sections. In this paper we present the first automated system for performing this decomposition. We compare the performance of our system with ground truth data and report favorable results.

## 1   Introduction

Potentially cancerous tissue samples are analyzed by staining them with a combination of two or more dyes. We consider the problem of recovering the amount of dye absorbed for each of the stains from a stack of hyperspectral images of the tissue sample. Since the exact spectral profile of the dyes varies from one experiment to the next and is not available to the pathologist, the problem is an instance of blind source separation. The problem is of special interest to clinical and research pathologists as the amount of dye absorbed by the sample is used to determine a quantitative estimate of the amount of cancerous cells present in the tissue.

The current state of the art solution requires an expert to hand click representative points in the tissue image to indicate "pure" dye spectra. This procedure requires human intervention and hence is time consuming and error prone.

In this paper we present the first system capable of performing this color decomposition in a fully automated manner. We also describe a novel procedure for acquiring the ground truth data and quantifying the performance of our system.

The organization of the paper is as follows. In section 2 we address the problem of image alignment in hyperspectral stacks. Section 3 presents the problem of color unmixing and proposes two unsupervised techniques as solutions. Data acquisition and experiments are discussed in Section 4. Section 5 summarizes the study and provides concluding remarks.

## 2  Multi-Spectral Alignment

Color unmixing is a challenging problem in itself, but it is complicated further by the practicalities of multispectral imaging: the component spectral images are usually misaligned, due to chromatic aberration and shifting of the stage. If the images comprising the spectral stack are out of alignment by as little as half a pixel, the estimated stain percentages at a given pixel can be altered drastically. This can result in large inaccuracies in the resulting cancer diagnosis.

Empirically, we have observed that the misalignments between images in the spectral stack can be modeled as small affine transforms, i.e. global translation, stretching, and rotation. Letting $I(\mathbf{x})$ and $J(\mathbf{x})$ denote two images, where $\mathbf{x} = (x, y)^\top$, this assumption is expressed as

$$J(A\mathbf{x} + \mathbf{d}) = I(\mathbf{x})$$

where $A$ is the $2 \times 2$ matrix of affine coefficients

$$A = \left[ \begin{array}{cc} a_{11} & a_{12} \\ a_{21} & a_{22} \end{array} \right]$$

and $\mathbf{d}$ is a 2D translation vector.

In the case of unimodal images, the iterative method of Shi and Tomasi [12] has been very successful for the estimation of differential (subpixel) affine transforms, e.g. from frame to frame in a video sequence. However, feeding cross-modal images directly to this algorithm is ineffective since they violatie the brightness constancy assumption [3]. We have observed, however, that the high spatial-frequency structures, e.g. edges and lines, tend to be consistent throughout the stack. This forms the basis of our alignment technique. We use the Shi-Tomasi algorithm on a band-pass filtered version of the images in the stack. To perform the filtering we apply a Laplacian of Gaussian (LoG) kernel [8], expressed as

$$h(\mathbf{x}) = \nabla^2 e^{-\|\mathbf{x}\|^2 / 2\sigma^2}$$

where $\sigma$ controls the width of the filter, to each image. The LoG kernel acts as a bandpass filter, suppressing constant regions and smooth shading, admitting edges and lines, and suppressing high frequency noise. We empirically determined the optimal parameters for the filtering to be $\sigma{=}0.5$ and a window size of 10 pixels. With this step used as preprocessing, Shi and Tomasi's algorithm is able to register this pair of images. An example of a synthesized color image composed of a 3D spectral stack is shown with and without this registration step in Figure 1; the blurring caused by misalignment and the subsequent sharpening resulting from registration is evident.

## 3  Color Unmixing

Once the registration problem is adequately addressed, we can proceed with the determination of stain concentrations. The problem in its full generality is an instance of the blind source separation problem. Given a spectral stack of $n_s$ images

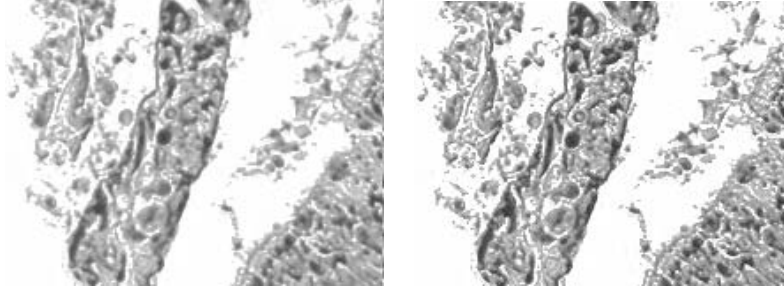

Figure 1: Synthesized color image representation of the same tissue core from a 10 dimensional spectral stack (a) with and (b) without differential affine registration.

obtained from imaging a tissue sample stained with $n_d$ dyes, with $n_s > n_d$, we wish to recover the staining due to each individual dye.

In an ideal world, the spectral profile of each dye would be exactly aligned with one of the spectral bands, and the absorptions measured therein would directly yield the stain concentrations. Realistically, however, the spectral profile of the dyes overlap and extend over several spectral bands, and the goal of recovering the $n_d$ components representing the dye percentages requires more careful analysis.

The problem of unmixing the dyes can be formulated as a matrix factorization problem:

$$\mathbf{X} = \mathbf{A}\mathbf{S} \tag{1}$$

Here $\mathbf{X}$ is an $n_s \times l$ column matrix, where $l$ is the number of pixels and the entry $\mathbf{X}_{ij}$ is the brightness of the $i^{th}$ pixel in the image in to the $j^{th}$ spectral band. The matrix $\mathbf{A}$ is an $n_s \times n_d$ matrix where each column of the matrix corresponds to the one of the dyes used in staining the tissue. $\mathbf{S}$ is a $n_s \times l$ matrix, with the entry $\mathbf{S}_{ij}$ indicating the contribution of the $i^{th}$ dye to the $j^{th}$ pixel.

The current state of the art solution for this problem in the field of automated pathology is Color Deconvolution [11], which yields acceptable results, but requires manual interaction in the form of mouse clicks on seed colors for the dyes. This is an example of a supervised technique. However, given the data matrix $\mathbf{X}$, there are a number of ways in which Equation (1) can be solved in a completely automatic manner without any human intervention. The three main classes of such methods are Principal Component Analysis (PCA), Non-negative Matrix Factorization (NMF) and Independent Component Analysis (ICA).

In this work we assume that staining is an additive process. Once a part of a tissue has been stained with a dye, addition of another stain can only increase the staining. The additivity of the stains combined with the physical constraint that each dye color will have a non-negative response in each frequency band implies that $\mathbf{A}$ and $\mathbf{B}$ are forced to be restricted to the class of non-negative matrices.

Methods based on PCA work by enforcing orthogonality constraints on the columns of $\mathbf{A}$ and are not well suited for recovering the factorization $\mathbf{A}\mathbf{S}$. PCA depends heavily on cancellation effects, i.e. a balancing of positive and negative terms as occurs with Gibbs' phenomenon in Fourier series. This will result in PCA returning $\mathbf{A}$ and $\mathbf{S}$ with negative entries which have no physical basis. In the following we shall investigate the use of algorithms based on NMF and ICA.

## 3.1 Non-negative Matrix Factorization

NMF is in principle well suited to the task of color unmixing, as it finds a factorization of $\mathbf{X}$ into $\mathbf{A}$ and $\mathbf{S}$ such that

$$[\mathbf{A}, \mathbf{S}] = \underset{\mathbf{A}, \mathbf{S}}{\operatorname{argmin}} \|\mathbf{X} - \mathbf{A}\mathbf{S}\| \qquad (2)$$

subject to

$$\mathbf{A}_{ij} \geq 0, \quad \mathbf{S}_{ij} \geq 0$$

The above problem is underconstrained; it has a scale ambiguity. Given a solution $[\mathbf{A}, \mathbf{S}]$ of the above problem, $[\alpha\mathbf{A}, \mathbf{S}/\alpha]$ for $\alpha \neq 0$ is also a solution to this problem. We solve this problem by constraining each column of $\mathbf{A}$ to have unit norm. This does not affect the final solution, since only the proportion of each stain is needed in the final analysis; the exact intensity of the constituent stain is not important.

The choice of the norm $\|\cdot\|$ decides the particular algorithm used for performing the minimization. We have implemented an iterative algorithm for recovering the non-negative factorization of a matrix due to Seung & Lee [7]. We use the $L_2$ norm as a measure of the error.

## 3.2 Independent Component Analysis

An alternate approach to matrix factorization is Independent Component Analysis (ICA)[4]. While Non-negative Matrix Factorization is based on enforcing a non-negativity constraint, it says nothing about the image formation process. ICA is based on a generative view of the data, where the data is assumed to be a result of superpositioning a number of stochastically independent processes. In the case of histological staining, this corresponds to assuming that each dye stains the tissue independently of all the other dyes. The rows of the matrix $\mathbf{S}$ represent the individual stochastic processes and the columns of $\mathbf{A}$ code their interactions.

We implemented the Joint Approximate Diagonalization of Eigenmatrices (JADE) algorithm to recover the independent components of $\mathbf{X}$ [2]. This algorithm calculates the ICA decomposition of $\mathbf{X}$ by calculating the eigenvalue decomposition of the cumulant tensor of the data. The eigenvalues of cumulant tensor are vectors corresponding to the independent components of the mixture.

# 4 Experimental Results

## 4.1 Sample Preparation and Data Acqusition

The histologically stained tissues used in this study were derived from human biopsies. The tissues were fixed in Bouin's solution, and embedded in paraffin. Dewaxed tissue sections were exposed to polyclonal antibodies (PAB) generated against synthetic peptides and confirmed to be specific for the proteins of interest. The sections were stained using a diaminobenzidine (DAB)-based detection method employing the Envision-Plus-Horseradish Peroxidase (HRP) system using an automated staining technique [5, 6]. The DAB immunohistochemistry stain used for the tissue samples shown here covers the majority of the visible range of the color spectra under the transmission of white light.

Great care must be taken in the acquisition of color images since the extraction of spectral information is highly dependent on the quality of the raw data. Hyperspectral imaging has been shown to be the best means of doing so.

A spectral image stack can be acquired using a number of different approaches. We use a setup based on a set of fixed bandpass filters. The filters are placed in the optical path of the light in front of the light source or camera and transmit only the desired wavelength bands.

In the following experiments the images were acquired on a scanning cytometer [9, 10, 1] with a 20x, 0.5 NA Fluor Nikon objective lens using a set of 10 equally spaced band pass filters ranging from 413 nm to 663 nm. The dynamic range of each of the spectral bands was maximized by controlling the gain and the exposure of the imaging system. This is required to ensure an accurate hyperspectral-to-RGB reconstruction for result visualization. It is important to note that the gain and exposure coefficients were inverted prior to the unmixing as they have no bearing on the staining process.

In order to quantitatively evaluate the decomposition provided by NMF and ICA, we prepared a set of ground truth data using the following procedure. Using a set of four tissue samples, we first applied the DAB stain and captured the hyperspectral image stack. We then added the hematoxylin stain and acquired a second image stack. The second stack serves as the input to our algorithm and the resulting decomposition, which estimates the DAB staining, is compared with the first stack, which serves as the ground truth.

We now experimentally evaluate the use of NMF and ICA for the color decomposition problem. While reconstruction error represents a simple quantitative measure, it does not provide a standard for judging how accurately the estimated components represent the dye concentrations. We quantify the performance by comparing the ground truth single-stained image to the corresponding automatically extracted component of the doubly-stained tissue sample. Figure 2 reports the performance of the two algorithms. The error measure used is

$$\text{error} = 100 \times \frac{\sum_i (I_i - \hat{I}_i)^2}{\sum_i I_i^2} \tag{3}$$

where the sum is over all pixels, and $I_i$ and $\hat{I}_i$ denote the ground truth and the estimate, respectively. Figure 3 shows the results of applying NMF and ICA to an image patch.

|         | NMF   | ICA   |
|---------|-------|-------|
| set1    | 18.15 | 12.81 |
| set2    | 18.79 | 14.99 |
| set3    | 4.47  | 19.42 |
| set4    | 5.04  | 18.12 |
| overall | 12.65 | 18.75 |

Figure 2: This table shows the percent error for the two unmixing algorithms across the four image sets. The four sets of images are available at http://vision.ucsd.edu/.

## 5 Discussion

The above experiments indicate that both NMF and ICA are capable of performing color decomposition of tissue samples stained with multiple histological dyes. However, there remain a number of sources of error, both during image acquisition as well as in the decomposition stage. These include errors due to imperfect focussing

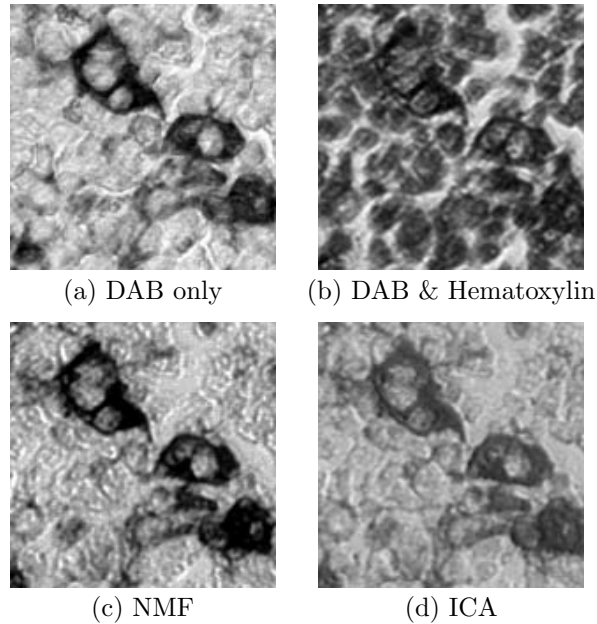

|  |  |
|---|---|
| (a) DAB only | (b) DAB & Hematoxylin |
| (c) NMF | (d) ICA |

Figure 3: Color unmixing using Non-negative Matrix factorization and Independent Component Analysis. Figure (a) shows a segment of the tissue stained using DAB, (b) shows the same tissue segment with DAB and Hematoxylin staining. The image in figure (b) serves as input to the two unmixing algorithms, the output of which is shown in (c) and (d). Figure (c) shows the DAB stain estimate produced by NMF and (d) shows the DAB staining estimated by ICA

in the various spectral bands and distortion in the acquired images which cannot be accounted for by optical flow based alignment methods such as Shi & Tomasi's algorithm. The principal source of discrepancy between the decomposition and the ground truth images, however, is caused by the chemical interaction between the various dyes used for staining. Measurement error due to dye interaction can be as high as 15%[13]. In this light, both ICA and NMF provide good results, and we expect that improvements in the image acquisition and registration procedure will result in systems capable of delivering performance close to the theoretical optimum.

In conclusion, we have addressed the problem of image registration for the planes in a hyperspectral stack for spectral information extraction and we proposed the use of two unsupervised algorithms, Non-negative Matrix Factorization and Independent Component Analysis, for extracting the contributions of various histological stains to the overall spectral composition throughout the tissue sample. We demonstrate the performance of these algorithms by comparing them with ground truth data.

We intend to address errors in the image acquisition and registration to further reduce the decomposition error in future work.

## References

[1] M. Bravo-Zanoguera, B. V. Massenbach, A. L. Kellner, and J. H. Price. High-performance autofocus circuit for biological microscopy. *Review of Scientific Instruments*, 69(11):3966–3977, 1998.

[2] Jean-Fracois Cardoso and Antoine Souloumiac. Blind beamforming for non gaussian signals. *IEE Proceedings-F*, 140(6), December 1993.

[3] B. K. P. Horn and B. G. Schunck. Determining optical flow. *Artificial Intelligence*, 17:185–204, 1981.

[4] A. Hyvärinen, J. Karhunen, and E. Oja. *Independent Component Analysis*. John Wiley & Sons, 2001.

[5] S. Krajewski, M. Krajewska, L.M. Ellerby, K. Welsh, Z. Xie, Q.L. Deveraux, G.S. Salvesen, D.E. Bredesen, R.E. Rosenthal, G. Fiskum, and J.C. Reed. Release of caspase - 9 from mitochondria during neuronal apoptosis and cerebral ischemia. *Proc Natl Acad Sci USA*, 96:5752–5757, 1999.

[6] S. Krajewski, M. Krajewska, A. Shabaik, T. Miyashita, H.G. Wang, and J.C. Reed. Immunohistochemical determination of in vivo distribution of Bax, a dominant inhibitor of Bcl - 2. *American Journal of Pathology*, 145:1323–1236, 1994.

[7] D. D. Lee and H. S. Seung. Learning the parts of objects with nonnegative matrix factorization. *Nature*, 401:788–791, 1999.

[8] David Marr. *Vision: A Computational Investigation into the Human Representation and Processing of Visual Information*. W. H. Freeman & Co., 1983.

[9] J. H. Price. *Scanning cytometry for cell monolayers*. PhD thesis, University of California, San Diego, 1990.

[10] J. H. Price, E. A. Hunter, and D. A. Gough. Accuracy of least squares designed spatial fir filters for segmentation of images of flourescence stained cell nuclei. *Cytometry*, 25:303–316, 1996.

[11] Arnout C. Ruifrok and Dennis A. Johnston. Quantification of histochemical staining by color deconvolution. *Analyt Quant Cytol Histol*, 23:291–299, 2001.

[12] Jianbo Shi and Carlo Tomasi. Good features to track. In *Proc. IEEE Conf. Comput. Vision and Pattern Recognition*, pages 593–600, 1994.

[13] R. J. Wordinger, G. W. Miller, and D. S. Nicodemus, editors. *Manual of Immunoperoxidase Techniques*. Americal Society of Clinical Pathologists, 1985.
